# Segmentation as Maximum-Weight Independent Set

**William Brendel and Sinisa Todorovic**
School of Electrical Engineering and Computer Science
Oregon State University
Corvallis, OR 97331
`brendelw@onid.orst.edu, sinisa@eecs.oregonstate.edu`

## Abstract

Given an ensemble of distinct, low-level segmentations of an image, our goal is to identify visually "meaningful" segments in the ensemble. Knowledge about any specific objects and surfaces present in the image is not available. The selection of image regions occupied by objects is formalized as the maximum-weight independent set (MWIS) problem. MWIS is the heaviest subset of mutually non-adjacent nodes of an attributed graph. We construct such a graph from all segments in the ensemble. Then, MWIS selects maximally distinctive segments that together partition the image. A new MWIS algorithm is presented. The algorithm seeks a solution directly in the discrete domain, instead of relaxing MWIS to a continuous problem, as common in previous work. It iteratively finds a candidate discrete solution of the Taylor series expansion of the original MWIS objective function around the previous solution. The algorithm is shown to converge to an optimum. Our empirical evaluation on the benchmark Berkeley segmentation dataset shows that the new algorithm eliminates the need for hand-picking optimal input parameters of the state-of-the-art segmenters, and outperforms their best, manually optimized results.

## 1    Introduction

This paper presents: (1) a new formulation of image segmentation as the maximum-weight independent set (MWIS) problem; and (2) a new algorithm for solving MWIS.

Image segmentation is a fundamental problem, and an area of active research in computer vision and machine learning. It seeks to group image pixels into visually "meaningful" segments, i.e., those segments that are occupied by objects and other surfaces occurring in the scene. The literature abounds with diverse formulations. For example, normalized-cut [1], and dominant set [2] formulate segmentation as a combinatorial optimization problem on a graph representing image pixels. "Meaningful" segments may give rise to modes of the pixels' probability distribution [3], or minimize the Mumford-Shah energy [4]. Segmentation can also be done by: (i) integrating edge and region detection [5], (ii) learning to detect and close object boundaries [6, 7], and (iii) identifying segments which can be more easily described by their own parts than by other image parts [8, 9, 10].

From prior work, we draw the following two hypotheses. First, surfaces of real-world objects are typically made of a unique material, and thus their corresponding segments in the image are characterized by unique photometric properties, distinct from those of other regions. To capture this distinctiveness, it seems beneficial to use more expressive, mid-level image features (e.g., superpixels, regions) which will provide richer visual information for segmentation, rather than start from pixels. Second, it seems that none of a host of segmentation formulations are able to correctly delineate every object boundary present. However, an ensemble of distinct segmentations is likely to contain a subset of segments that provides accurate spatial support of object occurrences. Based on these two hypotheses, below, we present a new formulation of image segmentation.

Given an ensemble of segments, extracted from the image by a number of different low-level segmenters, our goal is to select those segments from the ensemble that are distinct, and together partition the image area. Suppose all segments from the ensemble are represented as nodes of a graph, where node weights capture the distinctiveness of corresponding segments, and graph edges connect nodes whose corresponding segments overlap in the image. Then, the selection of maximally distinctive and non-overlapping segments that will partition the image naturally lends itself to the maximum-weight independent set (MWIS) formulation.

The MWIS problem is to find the heaviest subset of mutually non-adjacent nodes of an attributed graph. It is a well-researched combinatorial optimization problem that arises in many applications. It is known to be NP-hard, and hard to approximate [11]. Numerous heuristic approaches exist. For example, iterated tabu search [12] and branch-and-price [13] use a trial-and-error, greedy search in the space of possible solutions, with an optimistic complexity estimate of $O(n^3)$, where $n$ is the number of nodes in the graph. The message passing [14] relaxes MWIS into a linear program (LP), and solves it using loopy belief propagation with no guarantees of convergence for general graphs; the "tightness" of this relaxation holds only for bipartite graphs [15]. The semi-definite programming formulation of MWIS [16] provides an upper bound of the sum of weights of all independent nodes in MWIS. However, this is done by reformulating MWIS as a large LP of a new graph with $n^2$ nodes, which is unsuitable for large-scale problems as ours. Finally, the replicator dynamics [17, 18] converts the original graph into its complement, and solves MWIS as a continuous relaxation of the maximum weight clique (MWC) problem. But in some domains, including ours, important hard constraints captured by edges of the original graph may be lost in this conversion.

In this paper, we present a new MWIS algorithm, which represents a fixed-point iteration, guaranteed to converge to an optimum. It goes back and forth between the discrete and continuous domains. It visits a sequence of points $\{\boldsymbol{y}^{(t)}\}_{t=1,2,\dots}$, defined in the continuous domain, $\boldsymbol{y}^{(t)}\in[0,1]^n$. Around each of these points, the algorithm tries to maximize the objective function of MWIS in the discrete domain. Each iteration consists of two steps. First, we use the Taylor expansion to approximate the objective function around $\boldsymbol{y}^{(t)}$. Maximization in the discrete domain of the approximation gives a candidate discrete solution, $\tilde{\boldsymbol{x}}\in\{0,1\}^n$. Second, if $\tilde{\boldsymbol{x}}$ increases the original objective, then this candidate is taken as the current solution $\tilde{\boldsymbol{x}}$, and the algorithm visits that point in the next iteration, $\boldsymbol{y}^{(t+1)}=\tilde{\boldsymbol{x}}$; else, the algorithm visits the interpolation point, $\boldsymbol{y}^{(t+1)}=\boldsymbol{y}^{(t)}+\eta(\tilde{\boldsymbol{x}}-\boldsymbol{y}^{(t)})$, which can be shown to be a local maximizer of the original objective for a suitably chosen $\eta$. The algorithm always improves the objective, finally converging to a maximum. For non-convex objective functions, our method tends to pass either through or near discrete solutions, and the best discrete one $\boldsymbol{x}^*$ encountered along the path is returned. Our algorithm has relatively low complexity, $O(|E|)$, where, in our case, $|E|\ll n^2$ is the number of edges in the graph, and converges in only a few steps.

**Contributions:** To the best of our knowledge, this paper presents the first formulation of image segmentation as MWIS. We derive a new MWIS algorithm that has low complexity, and prove that it converges to a maximum. Selecting segments from an ensemble so they cover the entire image and minimize a total energy has been used for supervised object segmentation [19]. They estimate "good" segments by using classifiers of a pre-selected number of object classes. In contrast, our input, and our approach are genuinely low-level, i.e., agnostic about any particular objects in the image. Our MWIS algorithm has lower complexity, and is arguably easier to implement than the dual decomposition they use for energy minimization. Our segmentation outperforms the state of the art on the benchmark Berkeley segmentation dataset, and our MWIS algorithm runs faster and yields on average more accurate solutions on benchmark datasets than other existing MWIS algorithms.

**Overview:** Our approach consists of the following steps (see Fig.1). *Step 1:* The image is segmented using a number of different, off-the-shelf, low-level segmenters, including meanshift [3], Ncuts [1], and gPb-OWT-UCM [7]. Since the right scale at which objects occur in the image is unknown, each of these segmentations is conducted at an exhaustive range of scales. *Step 2:* The resulting segments are represented as nodes of a graph whose edges connect only those segments that (partially) overlap in the image. A small overlap between two segments, relative to their area, may be ignored, for robustness. A weight is associated with each node capturing the distinctiveness of the corresponding segment from the others. *Step 3:* We find the MWIS of this graph. *Step 4:* The segments selected in the MWIS may not be able to cover the entire image, or may slightly overlap (holes and overlaps are marked red in Fig.1). The final segmentation is obtained by using standard morphological operators on region boundaries to eliminate these holes and overlaps. Note that there is no need for Step 4 if

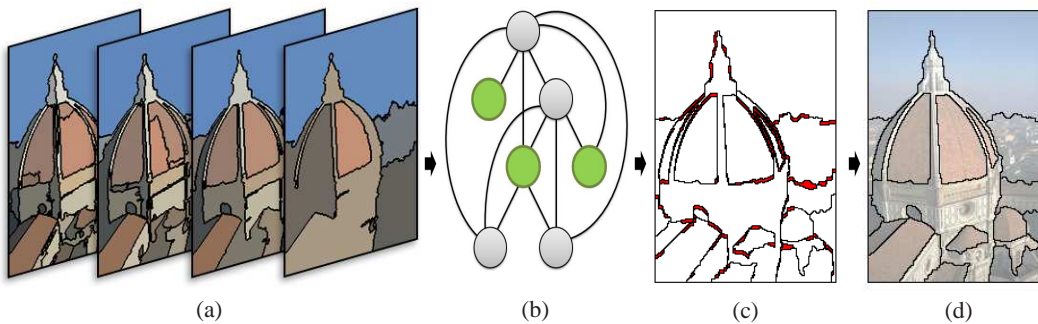

<div style="text-align:center">(a)          (b)          (c)          (d)</div>

Figure 1: Our main steps: (a) Input segments extracted at multiple scales by different segmentation algorithms; (b) Constructing a graph of all segments, and finding its MWIS (marked green); (c) Segments selected by our MWIS algorithm (red areas indicate overlaps and holes); (d) Final segmentation after region-boundary refinement (actual result using Meanshift and NCuts as input).

the input low-level segmentation is strictly hierarchical, as gPb-OWT-UCM [7]. The same holds if we added the intersections of all input segments to the input ensemble, as in [19], because our MWIS algorithm will continue selecting non-overlapping segments until the entire image is covered.

**Paper Organization:** Sec. 2 formulates MWIS, and presents our MWIS algorithm and its theoretical analysis. Sec. 3 formulates image segmentation as MWIS, and describes how to construct the segmentation graph. Sec. 4 and Sec. 5 present our experimental evaluation and conclusions.

## 2 MWIS Formulation and Our Algorithm

Consider a graph $G = (V, E, \omega)$, where $V$ and $E$ are the sets of nodes and undirected edges, with cardinalities $|V|=n$ and $|E|$, and $\omega : V \rightarrow \mathbb{R}^+$ associates positive weights $w_i$ to every node $i \in V$, $i=1,\ldots,n$. A subset of $V$ can be represented by an indicator vector $\boldsymbol{x}=(x_i)\in\{0,1\}^n$, where $x_i=1$ means that $i$ is in the subset, and $x_i=0$ means that $i$ is not in the subset. A subset $\boldsymbol{x}$ is called an independent set if no two nodes in the subset are connected by an edge, $\forall(i,j)\in E : x_i x_j=0$. We are interested in finding a maximum-weight independent set (MWIS), denoted as $\boldsymbol{x}^*$. MWIS can be naturally posed as the following integer program (IP):

$$\text{IP:} \quad \begin{aligned} \boldsymbol{x}^* &= \operatorname{argmax}_{\boldsymbol{x}} \boldsymbol{w}^{\mathrm{T}}\boldsymbol{x}, \\ \text{s.t. } &\forall i \in V\colon x_i \in \{0,1\}, \text{ and } \forall(i,j)\in E\colon x_i x_j = 0 \end{aligned} \quad (1)$$

The non-adjacency constraint in (1) can be equivalently formalized as $\sum_{(i,j)\in E} x_i x_j=0$. The latter expression can be written as a quadratic constraint, $\boldsymbol{x}^{\mathrm{T}}A\boldsymbol{x}=0$, where $\mathbf{A}=(\mathbf{A}_{ij})$ is the adjacency matrix, with $\mathbf{A}_{ij}=1$ if $(i,j)\in E$, and $\mathbf{A}_{ij}=0$ if $(i,j)\notin E$. Consequently, IP can be reformulated as the following integer quadratic program (IQP):

$$\begin{aligned} \boldsymbol{x}^* &= \operatorname{argmax}_{\boldsymbol{x}} \boldsymbol{w}^{\mathrm{T}}\boldsymbol{x}, \\ \text{s.t. } &\forall i \in V\colon x_i \in \{0,1\}, \ \boldsymbol{x}^{\mathrm{T}}A\boldsymbol{x} = 0 \end{aligned} \quad \underset{\exists \alpha \in \mathbb{R}}{\Rightarrow} \quad \text{IQP: } \begin{aligned} \boldsymbol{x}^* &= \operatorname{argmax}_{\boldsymbol{x}}[\boldsymbol{w}^{\mathrm{T}}\boldsymbol{x} - \tfrac{1}{2}\alpha\boldsymbol{x}^{\mathrm{T}}A\boldsymbol{x}] \\ \text{s.t. } &\forall i \in V\colon x_i \in \{0,1\} \end{aligned} \quad (2)$$

where there exists a positive regularization parameter $\alpha>0$ such that the problem on the implication in (2) holds. Next, we present our new algorithm for solving MWIS.

### 2.1 The Algorithm

As reviewed in Sec. 1, to solve IQP in (2), the integer constraint is usually either ignored, or relaxed to a continuous QP, e.g., by $\forall i\in V\colon x_i\geq 0$ and $\|\boldsymbol{x}\|=1$. For example, when $\ell_1$ norm is used as relaxation, the solution $\boldsymbol{x}^*$ of (2) can be found using the replicator dynamics in the continuous domain [17]. Also, when only $\forall i\in V\colon x_i\geq 0$ is used as relaxation, then the IP of (1) can be solved via message passing [14]. Usually, the solution found in the continuous domain is binarized to obtain a discrete solution. This may lead to errors, especially if the relaxed QP is nonconvex [20]. In this paper, we present a new MWIS algorithm that iteratively seeks a solution directly in the discrete domain. A discrete solution is computed by maximizing the first-order Taylor series approximation

of the quadratic objective in (2) around a solution found in the previous iteration. This is similar to the method of [20], which, however, makes the restrictive assumptions that the matrix of the quadratic term (analog of our $\mathbf{A}$) is "close" to positive-semi-definite (PSD), or that it is rank-1 with non-negative elements. These assumptions are not suitable for image segmentation. Graduated assignment [21] also iteratively maximizes a Taylor series expansion of a continuous QP around the previous solution; but this is done in the continuous domain. Since $\mathbf{A}$ in (2) is not PSD, our algorithm guarantees convergence only to a local maximum, as most state-of-the-art MWIS algorithms [12, 13, 14, 17, 18]. Below, we describe the main steps of our MWIS algorithm.

Let $f(\boldsymbol{x}) = \boldsymbol{w}^{\mathrm{T}}\boldsymbol{x} - \frac{1}{2}\alpha\boldsymbol{x}^{\mathrm{T}}A\boldsymbol{x}$ denote the objective function of IQP in (2). Also, in our notation, $\boldsymbol{x}, \tilde{\boldsymbol{x}}, \boldsymbol{x}^* \in \{0,1\}^n$ denote a point, candidate solution, and solution, respectively, in the discrete domain; and $\boldsymbol{y} \in [0,1]^n$ denotes a point in the continuous domain. Our algorithm is a fixed-point iteration that solves a sequence of integer programs which are convex approximations of $f$, around a solution found in the previous iteration. The key intuition is that the approximations are simpler functions than $f$, and thus facilitate computing the candidate discrete solutions in each iteration. The algorithm increases $f$ in every iteration until convergence.

Our algorithm visits a sequence of continuous points $\{\boldsymbol{y}^{(1)}, \ldots, \boldsymbol{y}^{(t)}, \ldots\}$, $\boldsymbol{y}^{(t)} \in [0,1]^n$, in iterations $t = 1, 2, \ldots$, and finds discrete candidate solutions $\tilde{\boldsymbol{x}} \in \{0,1\}^n$ in their respective neighborhoods, until convergence. Each iteration $t$ consists of two steps. First, for any point $\boldsymbol{y} \in [0,1]^n$ in the neighborhood of $\boldsymbol{y}^{(t)}$, we find the first-order Taylor series approximation of $f(\boldsymbol{y})$ as

$$f(\boldsymbol{y}) \approx h(\boldsymbol{y}, \boldsymbol{y}^{(t)}) = f(\boldsymbol{y}^{(t)}) + (\boldsymbol{y} - \boldsymbol{y}^{(t)})^{\mathrm{T}}(\boldsymbol{w} - \alpha\mathbf{A}\boldsymbol{y}^{(t)}) = \boldsymbol{y}^{\mathrm{T}}(\boldsymbol{w} - \alpha\mathbf{A}\boldsymbol{y}^{(t)}) + \text{const}, \quad (3)$$

where 'const' does not depend on $\boldsymbol{y}$. Note that the approximation $h(\boldsymbol{y}, \boldsymbol{y}^{(t)})$ is convex in $\boldsymbol{y}$, and simpler than $f(\boldsymbol{y})$, which allows us to easily compute a discrete maximizer of $h(\cdot)$ as

$$\tilde{\boldsymbol{x}} = \underset{\boldsymbol{x} \in \{0,1\}^n}{\operatorname{argmax}} h(\boldsymbol{x}, \boldsymbol{y}^{(t)}) \quad \Leftrightarrow \quad \tilde{x}_i = \left\{ \begin{array}{ll} 1 & , \quad \text{if } i\text{th element of } (\boldsymbol{w} - \alpha\mathbf{A}\boldsymbol{y}^{(t)})_i \geq 0 \\ 0 & , \quad \text{otherwise.} \end{array} \right. \quad (4)$$

To avoid the trivial discrete solution, when $\tilde{\boldsymbol{x}} = \mathbf{0}$ we instead set $\tilde{\boldsymbol{x}} = [0, \ldots, 0, 1, 0, \ldots, 0]^{\mathrm{T}}$, with $\tilde{x}_i = 1$ where $i$ is the index of the minimum element of $(\boldsymbol{w} - \alpha\mathbf{A}\boldsymbol{y}^{(t)})$.

In the second step of iteration $t$, the algorithm verifies if $\tilde{\boldsymbol{x}}$ can be accepted as a new, valid discrete solution. This will be possible only if $f$ is non-decreasing, i.e., if $f(\tilde{\boldsymbol{x}}) \geq f(\boldsymbol{y}^{(t)})$. In this case, the algorithm visits point $\boldsymbol{y}^{(t+1)} = \tilde{\boldsymbol{x}}$, in the next iteration. In case $f(\tilde{\boldsymbol{x}}) < f(\boldsymbol{y}^{(t)})$, this means that there must be a local maximum of $f$ in the neighborhood of points $\boldsymbol{y}^{(t)}$ and $\tilde{\boldsymbol{x}}$. We estimate this local maximizer of $f$ in the continuous domain by linear interpolation, $\boldsymbol{y}^{(t+1)} = \boldsymbol{y}^{(t)} + \eta(\tilde{\boldsymbol{x}} - \boldsymbol{y}^{(t)})$. The optimal value of the interpolation parameter $\eta \in [0,1]$ is computed such that $\partial f(\boldsymbol{y}^{(t+1)})/\partial\eta \geq 0$, which ensures that $f$ is non-decreasing in the next iteration. As shown in Sec. 2.2, the optimal $\eta$ has a closed-form solution:

$$\eta = \min\left(\max\left(\frac{(\boldsymbol{w} - \alpha\mathbf{A}\boldsymbol{y}^{(t)})^{\mathrm{T}}(\tilde{\boldsymbol{x}} - \boldsymbol{y}^{(t)})}{\alpha(\tilde{\boldsymbol{x}} - \boldsymbol{y}^{(t)})^{\mathrm{T}}\mathbf{A}(\tilde{\boldsymbol{x}} - \boldsymbol{y}^{(t)})}, 0\right), 1\right). \quad (5)$$

Having computed $\boldsymbol{y}^{(t+1)}$, the algorithm starts the next iteration by finding a Taylor series approximation in the neighborhood of point $\boldsymbol{y}^{(t+1)}$. After convergence, the latest discrete solution $\tilde{\boldsymbol{x}}$ is taken to represent the final solution of MWIS, $\boldsymbol{x}^* = \tilde{\boldsymbol{x}}$. Our MWIS algorithm is summarized in Alg. 1

## 2.2 Theoretical Analysis

This section presents the proof that our MWIS algorithm converges to a maximum. We also show that its complexity is $O(|E|)$. We begin by stating a lemma that pertains to linear interpolation $\boldsymbol{y}^{(t+1)} = \boldsymbol{y}^{(t)} + \eta(\tilde{\boldsymbol{x}} - \boldsymbol{y}^{(t)})$ such that the IQP objective function $f$ is non-decreasing at $\boldsymbol{y}^{(t+1)}$.

**Lemma 1** *Suppose that the IQP objective function $f$ is increasing at point $\boldsymbol{y}_1 \in [0,1]^n$, and decreasing at point $\boldsymbol{y}_2 \in [0,1]^n$, $\boldsymbol{y}_1 \neq \boldsymbol{y}_2$. Then, there exists a point, $\boldsymbol{y} = \boldsymbol{y}_1 + \eta(\boldsymbol{y}_2 - \boldsymbol{y}_1)$, and $\boldsymbol{y} \in [0,1]^n$, such that $f$ is increasing at $\boldsymbol{y}$, where $\eta$ is an interpolation parameter, $\eta \in [0,1]$.*

**Proof:** It is straightforward to show that if $\eta \in [0,1] \Rightarrow \boldsymbol{y} \in [0,1]^n$. For $\eta = 0$, we obtain $\boldsymbol{y} = \boldsymbol{y}_1$, where $f$ is said to be increasing. For $\eta \neq 0$, $\boldsymbol{y}$ can be found by estimating $\eta$ such

that $\partial f\big(\boldsymbol{y}_1+\eta(\boldsymbol{y}_2-\boldsymbol{y}_1)\big)/\partial\eta\geq0$. It follows: $(\boldsymbol{w}-\alpha\mathbf{A}\boldsymbol{y}_1)^{\mathrm{T}}(\boldsymbol{y}_2-\boldsymbol{y}_1)-\eta\alpha(\boldsymbol{y}_2-\boldsymbol{y}_1)^{\mathrm{T}}\mathbf{A}(\boldsymbol{y}_2-\boldsymbol{y}_1)\geq0$. Define auxiliary terms $c=(\boldsymbol{w}-\alpha\mathbf{A}\boldsymbol{y}_1)^{\mathrm{T}}(\boldsymbol{y}_2-\boldsymbol{y}_1)$ and $d=\alpha(\boldsymbol{y}_2-\boldsymbol{y}_1)^{\mathrm{T}}\mathbf{A}(\boldsymbol{y}_2-\boldsymbol{y}_1)$. Since $A$ is not PSD, we obtain $\eta\leq\frac{c}{d}$, for $d>0$, and $\eta\geq\frac{c}{d}$, for $d<0$. Since $\eta\in[0,1]$, we compute $\eta=\min(\max(\frac{c}{d},0),1)$, which is equivalent to (5), for $\boldsymbol{y}_1=\boldsymbol{y}^{(t)}$ and $\boldsymbol{y}_2=\tilde{\boldsymbol{x}}$. $\square$

In the following, we define the notion of maximum, and prove that Alg. 1 converges to a maximum.

**Definition** We refer to point $\boldsymbol{y}^*$ as a maximum of a real, differentiable function $g(\boldsymbol{y})$, defined over domain $\mathcal{D}$, $g:\mathcal{D}\to\mathbb{R}$, if there exists a neighborhood of $\boldsymbol{y}^*$, $\mathcal{N}(\boldsymbol{y}^*)\subseteq\mathcal{D}$, such that $\forall\boldsymbol{y}\in\mathcal{N}(\boldsymbol{y}^*):\ g(\boldsymbol{y}^*)\geq g(\boldsymbol{y})$.

**Proposition 1** *Alg. 1 increases $f$ in every iteration, and converges to a maximum.*

**Proof:** In iteration $t$ of Alg. 1, if $f(\tilde{\boldsymbol{x}})\geq f(\boldsymbol{y}^{(t)})$ then the next point visited by Alg. 1 is $\boldsymbol{y}^{(t+1)}=\tilde{\boldsymbol{x}}$. Thus, $f$ increases in this case. Else, $\boldsymbol{y}^{(t+1)}=\boldsymbol{y}^{(t)}+\eta(\tilde{\boldsymbol{x}}-\boldsymbol{y}^{(t)})$, yielding

$$f(\boldsymbol{y}^{(t+1)})=f(\boldsymbol{y}^{(t)})+\eta(\boldsymbol{w}-\alpha\mathbf{A}\boldsymbol{y}^{(t)})^{\mathrm{T}}(\tilde{\boldsymbol{x}}-\boldsymbol{y}^{(t)})+\eta^2\frac{1}{2}\alpha(\tilde{\boldsymbol{x}}-\boldsymbol{y}^{(t)})^{\mathrm{T}}\mathbf{A}(\tilde{\boldsymbol{x}}-\boldsymbol{y}^{(t)}). \quad (6)$$

Since $\tilde{\boldsymbol{x}}$ maximizes $h$, given by (3), we have $h(\tilde{\boldsymbol{x}},\boldsymbol{y}^{(t)})-h(\boldsymbol{y}^{(t)},\boldsymbol{y}^{(t)})=(\boldsymbol{w}-\alpha\mathbf{A}\boldsymbol{y}^{(t)})^{\mathrm{T}}(\tilde{\boldsymbol{x}}-\boldsymbol{y}^{(t)})\geq0$. Also, from Lemma 1, $\eta$ is non-negative. Consequently, the second term in (6) is non-negative. Regarding the third term in (6), from (5) we have $\eta\alpha(\tilde{\boldsymbol{x}}-\boldsymbol{y}^{(t)})^{\mathrm{T}}\mathbf{A}(\tilde{\boldsymbol{x}}-\boldsymbol{y}^{(t)})=(\boldsymbol{w}-\alpha\mathbf{A}\boldsymbol{y}^{(t)})^{\mathrm{T}}(\tilde{\boldsymbol{x}}-\boldsymbol{y}^{(t)})$ which we have already proved to be non-negative. Thus, $f$ also increases in this second case. Since $f\leq\boldsymbol{w}^{\mathrm{T}}\mathbf{1}$, and $f$ increases in every iteration, then $f$ converges to a maximum. $\square$

**Complexity:** Alg. 1 has complexity $O(|E|)$ per iteration. Complexity depends only on a few matrix-vector multiplications with $\mathbf{A}$, where each takes $O(|E|)$. This is because $\mathbf{A}$ is sparse and binary, where each element $\mathbf{A}_{ij}=1$ iff $(i,j)\in E$. Thus, any computation in Alg. 1 pertaining to particular node $i\in V$ depends on the number of positive elements in $i$th row $\mathbf{A}_{i\cdot}$, i.e., on the branching factor of $i$. Computing $\tilde{\boldsymbol{x}}$ in (4) has complexity $O(n)$, where $n<|E|$, and thus does not affect the final complexity. For the special case of balanced graphs, Alg. 1 has complexity $O(|E|)=O(n\log n)$. In our experiments, Alg. 1 converges in 5-10 iterations on graphs with about 300 nodes.

## 3 Formulating Segmentation as MWIS

We formulate image segmentation as the MWIS of a graph of image regions obtained from different segmentations. Below, we explain how to construct this graph. Given a set of all segments, $V$, extracted from the image by a number of distinct segmenters, we construct a graph, $G=(V,E,\omega)$, where $V$ and $E$ are the sets of nodes and undirected edges, and $\omega:V\to\mathbb{R}^+$ assigns positive weights $w_i$ to every node $i\in V$, $i=1,\ldots,n$. Two nodes $i$ and $j$ are adjacent, $(i,j)\in E$, if their respective segments $S_i$ and $S_j$ overlap in the image, $S_i\cap S_j\neq\emptyset$. This can be conceptualized by the adjacency matrix $\mathbf{A}=(\mathbf{A}_{ij})$, where $\mathbf{A}_{ij}=1$ iff $S_i\cap S_j\neq\emptyset$, and $\mathbf{A}_{ij}=0$ iff $S_i\cap S_j=\emptyset$. For robustness in our experiments, we tolerate a relatively small amount of overlap by setting a tolerance threshold $\theta$, such that $\mathbf{A}_{ij}=1$ if $\frac{|S_i\cap S_j|}{\min(|S_i|,|S_j|)}>\theta$, and $\mathbf{A}_{ij}=0$ otherwise. (In our experiments we use $\theta=0.2$). Note that the IQP in (2) also permits a "soft" definition of $\mathbf{A}$ which is beyond our scope.

The weights $w_i$ should be larger for more "meaningful" segments $S_i$, so that these segments are more likely included in the MWIS of $G$. Following the compositionality-based approaches of [8, 9], we define that a "meaningful" segment can be easily described in terms of its own parts, but difficult to describe via other parts of the image. Note that this definition is suitable for identifying both: (i) distinct textures in the image, since texture can be defined as a spatial repetition of elementary 2D patterns; and (ii) homogeneous regions with smooth variations of brightness. To define $w_i$, we use the formalism of [8], where the easiness and difficulty of describing $S_i$ is evaluated by its description length in terms of visual codewords. Specifically, given a dictionary of visual codewords, and the histogram of occurrence of the codewords in $S_i$, we define $w_i=|S_i|KL(S_i,\bar{S}_i)$, where $KL$ denotes the Kullback Leibler divergence, $I$ is the input image, and $\bar{S}_i=I\backslash S_i$. All the weights $\boldsymbol{w}$ are normalized by $\max_i w_i$. Below, we explain how to extract the dictionary of codewords.

Similar to [22], we describe every pixel with an 11-dimensional descriptor vector consisting of the *Lab* colors and filter responses of the rotationally invariant, nonlinear MR8 filter bank, along with

the Laplacian of Gaussian filters. The pixel descriptors are then clustered using K-means (with $K = 100$). All pixels grouped within one cluster are labeled with a unique codeword id of that cluster. Then, the histogram of their occurrence in every region $S_i$ is estimated.

Given $G$, as described in this section, we use our MWIS algorithm to select "meaningful" segments, and thus partition the image. Note that the selected segments will optimally cover the entire image, otherwise any uncovered image areas will be immediately filled out by available segments in $V$ that do not overlap with already selected ones, because this will increase the IQP objective function $f$. In the case when the input segments do not form a strict hierarchy and intersections of the input segments have not been added to $V$, we eliminate holes (or "soft" overlaps) between the selected segments by applying the standard morphological operations (e.g., thinning and dilating of regions).

## 4   Results

This section presents qualitative and quantitative evaluation of our segmentation on 200 images from the benchmark Berkeley segmentation dataset (BSD) [23]. BSD images are challenging for segmentation, because they contain complex layouts of distinct textures (e.g., boundaries of several regions meet at one point), thin and elongated shapes, and relatively large illumination changes. We also evaluate the generality and execution time of our MWIS algorithm on a synthetic graph from benchmark OR-Library [24], and the problem sets from [12].

Our MWIS algorithm is evaluated for the following three types of input segmentations. The first type is a hierarchy of segments produced by the gPb-OWT-UCM method of [7]. gPb-OWT-UCM uses the perceptual significance of a region boundary, $P_b \in [0, 100]$, as an input parameter. To obtain the hierarchy, we vary $P_b = 20:5:70$. The second type is a hierarchy of segments produced by the multiscale algorithm of [5]. This method uses pixel-intensity contrast, $\sigma \in [0, 255]$, as an input parameter. To obtain the hierarchy, we vary $\sigma = 30:20:120$. Finally, the third type is a union of NCut [1] and Meanshift [3] segments. Ncut uses one input parameter – namely, the total number of regions, $N$, in the image. Meanshift uses three input parameters: feature bandwidth $b_f$, spatial bandwidth $b_s$, and minimum region area $S_{\min}$. We vary these parameters as $N = 10:10:100$, $b_f = 5.5:0.5:8.5$, $b_s = 4:2:10$, and $S_{\min} = 100:200:900$. The variants [7]+Ours and [5]+Ours serve to test whether our approach is capable of extracting "meaningful" regions from a multiscale segmentation. The variant ([3]+[1])+Ours evaluates our hypothesis that reasoning over an ensemble of distinct segmentations improves each individual one.

Segmentation of BSD images is used for a comparison with replicator dynamics approach of [17], which transforms the MWIS problem into the maximum weight clique problem, and then relaxes it into a continuous problem, denoted as MWC. In addition, we also use data from other domains – specifically, OR-Library [24] and the problem sets from [12] – for a comparison with other state-of-the-art MWIS algorithms.

**Qualitative evaluation:**   Fig. 3 and Fig. 4 show the performance of our variant [7]+Ours on example images from BSD. Fig. 4 also shows the best segmentations of [7] and [25], obtained by an exhaustive search for the optimal values of their input parameters. As can be seen in Fig. 4, the method of [7] misses to segment the grass under the tiger, and oversegments the starfish and the camel, which we correct. Our approach eliminates the need for hand-picking the optimal input parameters in [7], and yields results that are good even in cases when objects have complex textures (e.g. tiger and starfish), or when the boundaries are blurred or jagged (e.g. camel).

**Quantitative evaluation:** Table 1 presents segmentations of BSD images using our three variants: [7]+Ours, [5]+Ours, and ([3]+[1])+Ours. We consider the standard metrics: Probabilistic Rand Index ($PRI$), and Variation of Information ($VI$) [26]. $PRI$ between estimated and ground-truth segmentations, $S$ and $G$, is defined as the sum of the number of pairs of pixels that have the same label in $S$ and $G$, and those that have different labels in both segmentations, divided by the total number of pairs of pixels. $VI$ measures the distance between $S$ and $G$ in terms of their average conditional entropy. $PRI$ should be large, and $VI$ small. For all variants of our approach, we run the MWIS algorithm 10 times, starting from different initial points, and report the average $PRI$ and $VI$ values. For [7], we report their best results obtained by an exhaustive search for the optimal value of their input parameter $P_b$. As can be seen, [7]+Ours does not hand-pick the optimal input parameters, and outperforms the best results of original [7]. Surprisingly, when working with

**Algorithm 1**: Our MWIS Algorithm

**Input**: Graph $G$ including $\boldsymbol{w}$ and $\mathbf{A}$, convergence threshold $\delta$, regularization parameter $\alpha = 2$

**Output**: The MWIS of $G$ denoted as $\boldsymbol{x}^*$

1   Define IQP objective: $f(\boldsymbol{x}) \triangleq \boldsymbol{w}^{\mathrm{T}}\boldsymbol{x} - \frac{1}{2}\alpha\boldsymbol{x}^{\mathrm{T}}A\boldsymbol{x}$ ;

2   Initialize $t{=}0$, and $\boldsymbol{x}^*{=}\boldsymbol{0}$, $\boldsymbol{y}^{(0)}{\in}\{0,1\}^n$, $\boldsymbol{y}^{(0)}{\neq}\boldsymbol{0}$;

3   **repeat**

4     Find $h(\boldsymbol{y}, \boldsymbol{y}^{(t)})$ as in (3);

5     Use (4) for $\tilde{\boldsymbol{x}}{=}\arg\max_{\boldsymbol{x}\in\{0,1\}^n} h(\boldsymbol{x}, \boldsymbol{y}^{(t)})$ ;

6     **if** $f(\tilde{\boldsymbol{x}}) \geq f(\boldsymbol{y}^{(t)})$ **then**

7       $\boldsymbol{y}^{(t+1)} = \tilde{\boldsymbol{x}}$ ;

8     **else**

9       Use (5) for
$$\eta = \underset{\eta\in[0,1]}{\arg\max}\, f\big(\boldsymbol{y}^{(t)}+\eta(\tilde{\boldsymbol{x}}-\boldsymbol{y}^{(t)})\big)$$

10       $\boldsymbol{y}^{(t+1)} = \boldsymbol{y}^{(t)} + \eta(\tilde{\boldsymbol{x}} - \boldsymbol{y}^{(t)})$ ;

11     **end**

12     **if** $f(\tilde{\boldsymbol{x}}) \geq f(\boldsymbol{x}^*)$ **then**

13       $\boldsymbol{x}^* = \tilde{\boldsymbol{x}}$ ;

14     **end**

15 **until** $\big\|\boldsymbol{y}^{(t+1)} - \boldsymbol{y}^{(t)}\big\| < \delta$ ;

| Method | $PRI$ | $VI$ |
|---|---|---|
| Human | 0.87 | 1.16 |
| [7] | 0.81 | 1.68 |
| ([3]+[1])+MWC | 0.78 | 1.75 |
| [5]+Ours | 0.79 | 1.69 |
| ([3]+[1])+Ours | 0.80 | 1.71 |
| [7]+Ours | 0.83 | 1.59 |

Table 1: A comparison on BSD. Probabilistic Rand Index ($PRI$) should be large, and Variation of Information ($VI$) small. Input segments are generated by the methods of [7, 5, 3, 1], and then selected by the maximum weight clique formulation (MWC) of [17], or by our algorithm. For [7], we report their best results obtained by an exhaustive search for the optimal value of their input parameter $P_b$.

segments generated by Meanshift, Ncuts, and [5], the performances of [5]+Ours and ([3]+[1])+Ours come very close to those of [7]. This is unexpected, because Meanshift, Ncuts, and the method of [5] are known to produce poor performance in terms of $PRI$ and $VI$ values, relative to [7]. Also, note that ([3]+[1])+Ours outperforms the relaxation-based method ([3]+[1])+MWC.

Fig. 2 shows the sensitivity of the convergence rate of our approach to a specific choice of $\alpha$. The penalty term $\alpha\boldsymbol{y}^{\mathrm{T}}A\boldsymbol{y}$ of the IQP objective function is averaged over all 200 graphs, each with about 300 nodes, obtained from 200 BSD images. As can be seen, for $\alpha \geq 2$, the penalty term $\alpha\boldsymbol{y}^{\mathrm{T}}A\boldsymbol{y}$ converges to 0 with some initial oscillations. Experimentally, the convergence rate is maximum when $\alpha = 2$. We use this value in all our experiments.

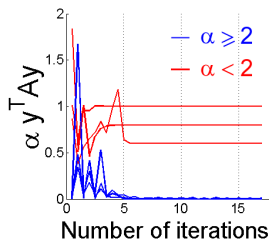

Figure 2: Convergence rate vs. a specific choice of $\alpha$, averaged over 200 BSD images: $\alpha < 2$ is marked red, and $\alpha \geq 2$ is marked blue.

| Method | | b2500 [24] | p3000-7000 [12] |
|---|---|---|---|
| [12] | avg | 2 | 175 |
| | sec | 74 | 1650 |
| Ours | avg | 0 | 62 |
| | sec | 21 | 427 |

Table 2: Average of solution difference, and computation time in seconds for problem sets from [24] and [12].

**MWIS performance:** We also test our Alg. 1 on two sets of problems beyond image segmentation. As input we use a graph constructed from data from the OR-Library [24], and from the problem sets presented in [12]. For the first set of problems (b2500), we only consider the largest graphs. We use ten instances, called b2500-1 to b2500-10, of size 2500 and with density 10%. For the second set of problem (p3000 to p7000), we take into account graphs of size 4000, 5000, 6000 and 7000. Five graph instances per size are used. Tab. 2 shows the average difference between the estimated and ground-truth solution, and computation time in seconds. The presented comparison with Iterative Tabu Search (ITS) [12] demonstrates that, on average, we achieve better performance, under much smaller running times.

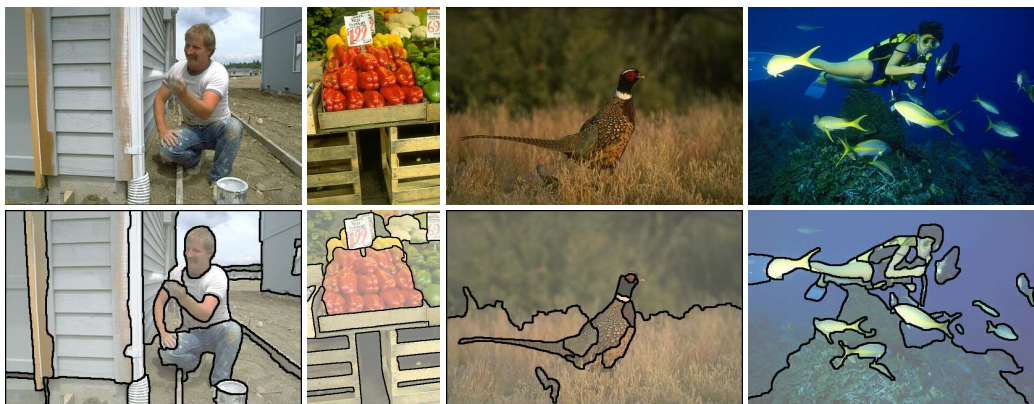

Figure 3: Segmentation of BSD images. (top) Original images. (bottom) Results using our variant [7]+Ours. Failures, such as the painters' shoulder, the bird's lower body part, and the top left fish, occur simply because these regions are not present in the input segmentations.

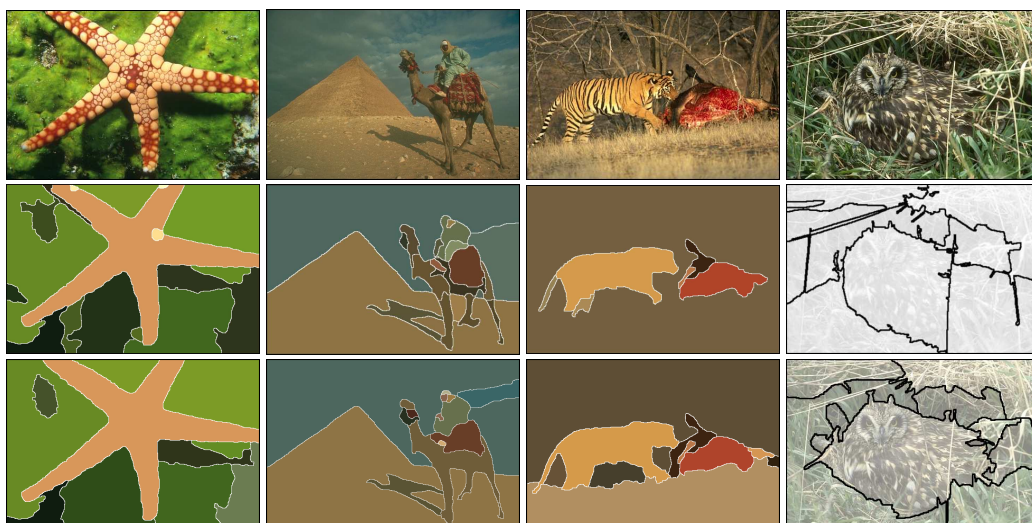

Figure 4: Comparison with the state-of-the-art segmentation algorithms on BSD images. (top row) Original images. (middle row) The three left results are from [7], and the rightmost result is from [25]. (bottom row) Results of [7]+Ours. By extracting "meaningful" segments from a segmentation hierarchy produced by [7] we correct the best, manually optimized results of [7].

## 5   Conclusion

To our knowledge, this is the first attempt to formulate image segmentation as MWIS. Our empirical findings suggest that this is a powerful framework that permits good segmentation performance regardless of a particular MWIS algorithm used. We have presented a new fixed point algorithm that efficiently solves MWIS, with complexity $O(|E|)$, on a graph with $|E|$ edges, and proved that the algorithm converges to a maximum. Our MWIS algorithm seeks a solution directly in the discrete domain, instead of resorting to the relaxation, as is common in the literature. We have empirically observed that our algorithm runs faster and outperforms the other competing MWIS algorithms on benchmark datasets. Also, we have shown a comparison with the state-of-the-art segmenter [7] on the benchmark Berkeley segmentation dataset. Our selection of "meaningful" regions from a segmentation hierarchy produced by [7] outperforms the manually optimized best results of [7], in terms of Probabilistic Rand Index and Variation of Information.

# References

[1] J. Shi and J. Malik, "Normalized cuts and image segmentation," *IEEE TPAMI*, vol. 22, no. 8, pp. 888–905, 2000.

[2] M. Pavan and M. Pelillo, "Dominant sets and pairwise clustering," *IEEE TPAMI*, vol. 29, no. 1, pp. 167–172, 2007.

[3] D. Comaniciu and P. Meer, "Meanshift: a robust approach toward feature space analysis," *IEEE TPAMI*, vol. 24, no. 5, pp. 603–619, 2002.

[4] M. Kass, A. Witkin, and D. Terzopoulos, "Snakes: Active contour models," *IJCV*, vol. V1, no. 4, pp. 321–331, 1988.

[5] N. Ahuja, "A transform for multiscale image segmentation by integrated edge and region detection," *IEEE TPAMI*, vol. 18, no. 12, pp. 1211–1235, 1996.

[6] X. Ren, C. Fowlkes, and J. Malik, "Learning probabilistic models for contour completion in natural images," *IJCV*, vol. 77, no. 1-3, pp. 47–63, 2008.

[7] P. Arbelaez, M. Maire, C. Fowlkes, and J. Malik, "From contours to regions: An empirical evaluation," in *CVPR*, 2009.

[8] S. Bagon, O. Boiman, and M. Irani, "What is a good image segment? A unified approach to segment extraction," in *ECCV*, 2008.

[9] S. Todorovic and N. Ahuja, "Texel-based texture segmentation," in *ICCV*, 2009.

[10] B. Russell, A. Efros, J. Sivic, B. Freeman, and A. Zisserman, "Segmenting scenes by matching image composites," in *NIPS*, 2009.

[11] L. Trevisan, "Inapproximability of combinatorial optimization problems," Electronic Colloquium on Computational Complexity, Tech. Rep. TR04065, 2004.

[12] G. Palubeckis, "Iterated tabu search for the unconstrained binary quadratic optimization problem," *Informatica*, vol. 17, no. 2, pp. 279–296, 2006.

[13] D. Warrier, W. E. Wilhelm, J. S. Warren, and I. V. Hicks, "A branch-and-price approach for the maximum weight independent set problem," *Netw.*, vol. 46, no. 4, pp. 198–209, 2005.

[14] S. Sanghavi, D. Shah, and A. S. Willsky, "Message-passing for max-weight independent set," in *NIPS*, 2007.

[15] M. Groetschel, L. Lovasz, and A. Schrijver, "Polynomial algorithms for perfect graphs," in *Topics on Perfect Graphs*, C. Berge and V. Chvatal, Eds. North-Holland, 1984, vol. 88, pp. 325 – 356.

[16] M. Todd, "Semidefinite optimization," *Acta Numerica*, vol. 10, pp. 515–560, 2001.

[17] I. M. Bomze, M. Pelillo, and V. Stix, "Approximating the maximum weight clique using replicator dynamics," *IEEE Trans. Neural Net.*, vol. 11, no. 6, pp. 1228–1241, 2000.

[18] S. Busygin, C. Ag, S. Butenko, and P. M. Pardalos, "A heuristic for the maximum independent set problem based on optimization of a quadratic over a sphere," *Journal of Combinatorial Optimization*, vol. 6, pp. 287–297, 2002.

[19] M. P. Kumar and D. Koller, "Efficiently selecting regions for scene understanding," in *CVPR*, 2010.

[20] M. Leordeanu, M. Hebert, and R. Sukthankar, "An integer projected fixed point method for graph matching and MAP inference," in *NIPS*, 2009.

[21] S. Gold and A. Rangarajan, "A graduated assignment algorithm for graph matching," *IEEE TPAMI*, vol. 18, no. 4, pp. 377–388, 1996.

[22] M. Varma and R. Garg, "Locally invariant fractal features for statistical texture classification," in *ICCV*, 2007.

[23] D. Martin, C. Fowlkes, D. Tal, and J. Malik, "A database of human segmented natural images and its application to evaluating segmentation algorithms and measuring ecological statistics," in *ICCV*, 2001.

[24] J. E. Beasley, "Obtaining test problems via internet," *Journal of Global Optimization*, vol. 8, no. 4, pp. 429–433, 1996.

[25] M. Galun, E. Sharon, R. Basri, and A. Brandt, "Texture segmentation by multiscale aggregation of filter responses and shape elements," in *ICCV*, 2003, pp. 716–723.

[26] R. Unnikrishnan, C. Pantofaru, and M. Hebert, "Toward objective evaluation of image segmentation algorithms," *IEEE TPAMI*, vol. 29, no. 6, pp. 929–944, 2007.

